# Integrating locally learned causal structures with overlapping variables

**Robert E. Tillman**
Carnegie Mellon University
Pittsburgh, PA 15213
rtillman@andrew.cmu.edu

**David Danks, Clark Glymour**
Carnegie Mellon University &
Institute for Human & Machine Cognition
Pittsburgh, PA 15213
{ddanks,cg09}@andrew.cmu.edu

## Abstract

In many domains, data are distributed among datasets that share only some variables; other recorded variables may occur in only one dataset. While there are asymptotically correct, informative algorithms for discovering causal relationships from a single dataset, even with missing values and hidden variables, there have been no such reliable procedures for distributed data with overlapping variables. We present a novel, asymptotically correct procedure that discovers a minimal equivalence class of causal DAG structures using local independence information from distributed data of this form and evaluate its performance using synthetic and real-world data against causal discovery algorithms for single datasets and applying Structural EM, a heuristic DAG structure learning procedure for data with missing values, to the concatenated data.

## 1 Introduction

In many domains, researchers are interested in predicting the effects of interventions, or manipulating variables, on other observed variables. Such predictions require knowledge of causal relationships between observed variables. There are existing asymptotically correct algorithms for learning such relationships from data, possibly with missing values and hidden variables [1][2][3], but these algorithms all assume that every variable is measured in a single study. Datasets for such studies are not always readily available, often due to privacy, ethical, financial, and practical concerns. However, given the increasing availability of large amounts of data, it is often possible to obtain several similar studies that individually measure subsets of the variables a researcher is interested in and together include all such variables. For instance, models of the United States and United Kingdom economies share some but not all variables, due to different financial recording conventions; fMRI studies with similar stimuli may record different variables, since the images vary according to magnet strength, data reduction procedures, etc.; and U.S. states report some of the same educational testing variables, but also report state-specific variables. In these cases, if each dataset has *overlapping variable(s)* with at least one other dataset, e.g. if two datasets $D_1$ and $D_2$, which measure variables $\mathbf{V}_1$ and $\mathbf{V}_2$, respectively, have at least one variable in common ($\mathbf{V}_1 \cap \mathbf{V}_2 \neq \emptyset$), then we should be able to learn many of the causal relationships between the observed variables using this set of datasets. The existing algorithms, however, cannot in general be directly applied to such cases, since they may require joint observations for variables that are not all measured in a single dataset.

While this problem has been discussed in [4] and [5], there are no general, useful algorithms for learning causal relationships from data of this form. A typical response is to concatenate the datasets to form a single common dataset with missing values for the variables that are not measured in each of the original datasets. Statistical matching [6] or multiple imputation [7] procedures may then be used to fill in the missing values by assuming an underlying model (or small class of models), estimating model parameters using the available data, and then using this model to interpolate the

missing values. While the assumption of some underlying model may be unproblematic in many standard prediction scenarios, i.e. classification, it is unreliable for causal inference; the causal relationships learned using the interpolated dataset that are between variables which are never jointly measured in single dataset will only be correct if the corresponding relationships between variables in the assumed model happen to be causal relationships in the correct model. The Structural EM algorithm [8] avoids this problem by iteratively updating the assumed model using the current interpolated dataset and then reestimating values for the missing data to form a new interpolated dataset until the model converges. The Structural EM algorithm is only justified, however, when missing data are missing at random (or indicator variables can be used to make them so) [8]. The pattern of missing values in the concatenated datasets described above is highly structured. Furthermore, Structural EM is a heuristic procedure and may converge to local maxima. While this may not be problematic in practice when doing prediction, it is problematic when learning causal relationships. Our experiments in section 4 show that Structural EM performs poorly in this scenario.

We present a novel, asymptotically correct algorithm—the *Integration of Overlapping Networks (ION) algorithm*—for learning causal relationships (or more properly, the complete set of possible causal DAG structures) from data of this form. Section 2 provides the relevant background and terminology. Section 3 discusses the algorithm. Section 4 presents experimental evaluations of the algorithm using synthetic and real-world data. Finally, section 5 provides conclusions.

## 2 Formal preliminaries

We now introduce some terminology. A *directed graph* $\mathcal{G} = \langle \mathcal{V}, \mathcal{E} \rangle$ is a set of nodes $\mathcal{V}$, which represent variables, and a set of directed edges $\mathcal{E}$ connecting distinct nodes. If two nodes are connected by an edge then the nodes are *adjacent*. For pairs of nodes $\{X, Y\} \subseteq \mathcal{V}$, $X$ is a parent (child) of $Y$, if there is a directed edge from $X$ to $Y$ ($Y$ to $X$) in $\mathcal{E}$. A *trail* in $\mathcal{G}$ is a sequence of nodes such that each consecutive pair of nodes in the sequence is adjacent in $\mathcal{G}$ and no node appears more than once in the sequence. A trail is a *directed path* if every edge between consecutive pairs of nodes points in the same direction. $X$ is an *ancestor* (*descendant*) of $Y$ if there is a directed path from $X$ to $Y$ ($Y$ to $X$). $\mathcal{G}$ is a *directed acyclic graph* (DAG) if for every pair $\{X, Y\} \subseteq \mathcal{V}$, $X$ is not both an ancestor and a descendent of $Y$ (no directed cycles). A *collider* (*v-structure*) is a triple of nodes $\langle X, Y, Z \rangle$ such that $X$ and $Z$ are parents of $Y$. A trail is *active* given $\mathbf{C} \subseteq \mathcal{V}$ if (i) for every collider $\langle X, Y, Z \rangle$ in the trail either $Y \in \mathbf{C}$ or some descendant of $Y$ is in $\mathbf{C}$ and (ii) no other node in the trail is in $\mathbf{C}$. For disjoint sets of nodes $\mathbf{X}$, $\mathbf{Y}$, and $\mathbf{Z}$, $\mathbf{X}$ is d-separated (d-connected) from $\mathbf{Y}$ given $\mathbf{Z}$ if and only if there are no (at least one) active trails between any $X \in \mathbf{X}$ and any $Y \in \mathbf{Y}$ given $\mathbf{Z}$.

A *Bayesian network* $\mathcal{B}$ is a pair $\langle \mathcal{G}, \mathcal{P} \rangle$, where $\mathcal{G} = \langle \mathcal{V}, \mathcal{E} \rangle$ is a DAG and $\mathcal{P}$ is a joint probability distribution over the variables represented by the nodes in $\mathcal{V}$ such that $\mathcal{P}$ can be decomposed as follows:

$$\mathcal{P}(\mathcal{V}) = \prod_{V \in \mathcal{V}} P(V | \mathbf{Parents}(V))$$

For $\mathcal{B} = \langle \mathcal{G}, \mathcal{P} \rangle$, if $\mathbf{X}$ is d-separated from $\mathbf{Y}$ given $\mathbf{Z}$ in $\mathcal{G}$, then $\mathbf{X}$ is conditionally independent of $\mathbf{Y}$ given $\mathbf{Z}$ in $\mathcal{P}$ [9]. For disjoint sets of nodes, $\mathbf{X}$, $\mathbf{Y}$, and $\mathbf{Z}$ in $\mathcal{V}$, $\mathcal{P}$ is *faithful* to $\mathcal{G}$ if $\mathbf{X}$ is d-separated from $\mathbf{Y}$ given $\mathbf{Z}$ in $\mathcal{G}$ whenever $\mathbf{X}$ is conditionally independent of $\mathbf{Y}$ given $\mathbf{Z}$ in $\mathcal{P}$ [1]. $\mathcal{B}$ is a *causal* Bayesian network if an edge from $X$ to $Y$ indicates that $X$ is a direct cause of $Y$ relative to $\mathcal{V}$.

Most algorithms for *causal discovery*, or learning causal relationships from nonexperimental data, assume that the distribution over the observed variables $\mathcal{P}$ is decomposable according to a DAG $\mathcal{G}$ and $\mathcal{P}$ is faithful to $\mathcal{G}$. The goal is to learn $\mathcal{G}$ using the data from $\mathcal{P}$. Most causal discovery algorithms return a set of possible DAGs which entail the same d-separations and d-connections, e.g. the *Markov equivalence class*, rather than a single DAG. The DAGs in this set have the same adjacencies but only some of the same directed edges. The directed edges common to each DAG represent causal relationships that are learned from the data. If we admit the possibility that there may be unobserved (latent) common causes between observed variables, then this set of possible DAGs is usually larger.

A *partial ancestral graph* (PAG) represents the set of DAGs in a particular Markov equivalence class when latent common causes may be present. Nodes in a PAG correspond to observed variables. Edges are of four types: $\rightarrow, \circ\!\!\rightarrow, \circ\!\!-\!\!\circ$ and $\leftrightarrow$, where a $\circ$ indicates either an $\blacktriangleright$ or $-$ orientation, bidirected edges indicate the presence of a latent common cause, and fully directed edges ($\rightarrow$)

indicate that the directed edge is present in every DAG, e.g. a causal relationship. For $\{X, Y\} \subseteq \mathcal{V}$, a *possibly* active trail between $X$ and $Y$ given $\mathbf{Z} \subseteq \mathcal{V}/\{X, Y\}$ is a trail in a PAG between $X$ and $Y$ such that some orientation of ∘'s on edges between consecutive nodes in the trail, to either $-$ or ▶, makes the trail active given $\mathbf{Z}$.

## 3 Integration of Overlapping Networks (ION) algorithm

The ION algorithm uses conditional independence information to discover the complete set of PAGs over a set of variables $\mathcal{V}$ that are consistent with a set of datasets over subsets of $\mathcal{V}$ which have overlapping variables. ION accepts as input a set of PAGs which correspond to each of such datasets. A standard causal discovery algorithm that checks for latent common causes, such as FCI [1] or GES [3] with latent variable postprocessing steps[1], must first be applied to each of the original datasets to learn these PAGs that will be input to ION. Expert domain knowledge can also be encoded in the input PAGs, if available. The ION algorithm is shown as algorithm 1 and described below.

---

**Input** : PAGs $\mathcal{G}_i \in \mathbf{G}$ with nodes $\mathbf{V}_i \subseteq \mathcal{V}$ for $i = 1, \ldots, k$
**Output**: PAGs $\mathcal{H}_i \in \mathbf{H}$ with nodes $\mathbf{V}_i = \mathcal{V}$ for $i = 1, \ldots, m$

1  $\mathcal{K} \leftarrow$ the complete graph over $\mathcal{V}$ with ∘'s at every endpoint
2  $\mathbf{A} \leftarrow \emptyset$
3  Transfer nonadjacencies and endpoint orientations from each $\mathcal{G}_i \in \mathbf{G}$ to $\mathcal{K}$ and propagate the changes in $\mathcal{K}$ using the rules described in [10]
4  $\mathbf{PAT}(\{\mathsf{X}, \mathsf{Y}\}, \mathbf{Z}) \leftarrow$ all *possibly* active trails between $\mathsf{X}$ and $\mathsf{Y}$ given $\mathbf{Z}$ for all $\{\mathsf{X}, \mathsf{Y}\} \subseteq \mathcal{V}$ and $\mathbf{Z} \subseteq \mathcal{V}/\{\mathsf{X}, \mathsf{Y}\}$ such that $\mathsf{X}$ and $\mathsf{Y}$ are d-separated given $\mathbf{Z}$ in some $\mathcal{G}_i \in \mathbf{G}$
5  $\mathbf{PC} \leftarrow$ all *minimal hitting sets* of changes to $\mathcal{K}$, such that all $\mathsf{PAT}_i \in \mathbf{PAT}$ are not active
6  **for** $\mathsf{PC}_i \in \mathbf{PC}$ **do**
7       $\mathcal{A}_i \leftarrow \mathcal{K}$ after making and propagating the changes $\mathsf{PC}_i$
8       **if** $\mathcal{A}_i$ *is consistent with every* $\mathcal{G}_i \in \mathbf{G}$ **then** add $\mathcal{A}_i$ to $\mathbf{A}$
9  **end**
10 **for** $\mathcal{A}_i \in \mathbf{A}$ **do**
11      Remove $\mathcal{A}_i$ from $\mathbf{A}$
12      Mark all edges in $\mathcal{A}_i$ as '?'
13      For each $\{\mathsf{X}, \mathsf{Y}\} \subseteq \mathcal{V}$ such that $\mathsf{X}$ and $\mathsf{Y}$ are adjacent in $\mathcal{A}_i$, if $\mathsf{X}$ and $\mathsf{Y}$ are d-connected given $\emptyset$ in some $\mathcal{G}_i \in \mathbf{G}$, then remove '?' from the edge between $\mathsf{X}$ and $\mathsf{Y}$ in $\mathcal{A}_i$
14      $\mathbf{PR} \leftarrow$ every combination of removing or not removing '?' marked edges from $\mathcal{A}_i$
15      **for** $\mathsf{PR}_i \in \mathbf{PR}$ **do**
16          $\mathcal{H}_i \leftarrow \mathcal{A}_i$ after making and propagating the changes $\mathsf{PR}_i$
17          **if** $\mathcal{H}_i$ *is consistent with every* $\mathcal{G}_i \in \mathbf{G}$ **then** add $\mathcal{H}_i$ to $\mathbf{H}$
18      **end**
19 **end**

**Algorithm 1**: The Integration of Overlapping Networks (ION) algorithm

---

The algorithm begins with the complete graph over $\mathcal{V}$ with all ∘ endpoints and transfers nonadjacencies and endpoint orientations from each $\mathcal{G}_i \in \mathbf{G}$ at line 3, e.g. if $X$ and $Y$ are not adjacent in $\mathcal{G}_i$ then remove the edge between $X$ and $Y$, if $X$ is directed into $Y$ in $\mathcal{G}_i$ then set the endpoint at $Y$ on the edge between $X$ and $Y$ to ▶. Once these orientations and edge removals are made, the changes to the complete graph are *propagated* using the rules in [10], which provably make every change that is entailed by the current changes made to the graph. Lines 4-9 find every possibly active trail for every $\{X, Y\} \subseteq \mathcal{V}$ given $\mathbf{Z} \subseteq \mathcal{V}/\{X, Y\}$ such that $X$ and $Y$ are d-separated given $\mathbf{Z}$ in some $\mathcal{G}_i \in \mathbf{G}$. The constructed set $\mathbf{PC}$ includes all *minimal hitting sets* of graphical changes, e.g. unique sets of minimal changes that are not subsets of other sets of changes, which make these paths no longer active. For each minimal hitting set, a new graph is constructed by making the changes in the set and propagating these changes. If the graph is consistent with each $\mathcal{G}_i \in \mathbf{G}$, e.g. the graph does not imply a d-separation for some $\{X, Y\} \subseteq \mathcal{V}$ given $\mathbf{Z} \subseteq \mathcal{V}/\{X, Y\}$ such that $X$ and $Y$ are d-connected in some $\mathcal{G}_i \in \mathbf{G}$, then this graph is added to the current set of possible graphs. Lines 10-

19 attempt to discover any additional PAGs that may be consistent with each $\mathcal{G}_i \in \mathbf{G}$ after deleting edges from PAGs in the current set and propagating the changes. If some pair of nodes $\{X, Y\} \subseteq \mathcal{V}$ that are adjacent in a current PAG are d-connected given $\emptyset$ in some $G_i \in \mathcal{G}$, then we do not consider sets of edge removals which remove this edge.

The ION algorithm is provably *sound* in the sense that the output PAGs are consistent with every $\mathcal{G}_i \in \mathbf{G}$, e.g. no $\mathcal{H}_i \in \mathbf{H}$ entails a d-separation or d-connection that contradicts a d-separation or d-connection entailed by some $\mathcal{G}_i \in \mathbf{G}$. This property follows from the fact that d-separation and d-connection are mutually exclusive, exhaustive relations.

**Theorem 3.1** (soundness). If $X$ and $Y$ are d-separated (d-connected) given $\mathbf{Z}$ in some $\mathcal{G}_i \in \mathbf{G}$, then $X$ and $Y$ are d-separated (d-connected) given $\mathbf{Z}$ in every $\mathcal{H}_i \in \mathbf{H}$.

*Proof Sketch.* Every structure $\mathcal{A}_i$ constructed at line 7 provably entails every d-separation entailed by some $\mathcal{G}_i \in \mathbf{G}$. Such structures are only added to $\mathbf{A}$ if they do not entail a d-separation corresponding to a d-connection in some $\mathcal{G}_i \in \mathbf{G}$. The only changes made (other than changes resulting from propagating other changes which are provably correct by [10]) in lines 10-19 are edge removals, which can only create new d-separations. If a new d-separation is created which corresponds to a d-connection in some $\mathcal{G}_i \in \mathbf{G}$, then the PAG entailing this new d-separation is not added to $\mathbf{H}$. □

The ION algorithm is provably *complete* in the sense that if there is some structure $\mathcal{H}_i$ over the variables $\mathcal{V}$ that is consistent with every $\mathcal{G}_i \in \mathbf{G}$, then $\mathcal{H}_i \in \mathbf{H}$.

**Theorem 3.2** (completeness). Let $\mathcal{H}_i$ be a PAG over the variables $\mathcal{V}$ such that for every pair $\{X, Y\} \subseteq \mathcal{V}$, if $X$ and $Y$ are d-separated (d-connected) given $\mathbf{Z} \subseteq \mathcal{V}/\{X, Y\}$ in some $\mathcal{G}_i \in \mathbf{G}$, then $X$ and $Y$ are d-separated (d-connected) given $\mathbf{Z}$ in $\mathcal{H}_i$. Then, $\mathcal{H}_i \in \mathbf{H}$.

*Proof Sketch.* Every change made at line 3 is provably necessary to ensure soundness. At least one graph added to $\mathbf{A}$ at line 8 provably has every adjacency (possibly more) in $\mathcal{H}_i$ and no non-∘ endpoints on an edge found in $\mathcal{H}_i$ that is not also present in $\mathcal{H}_i$. Some sequence of edge removals will provably produce $\mathcal{H}_i$ at line 16 and it will be added to the output set since it is consistent with every $\mathcal{G}_i \in \mathbf{G}$. □

Thus, by theorems 3.1 and 3.2, ION is an asymptotically correct algorithm for learning the complete set of PAGs over $\mathcal{V}$ that are consistent with a set of datasets over subsets of $\mathcal{V}$ with overlapping variables, if the input PAGs are discovered using an asymtotically correct algorithm that detects the presence of latent common causes, i.e. FCI, with each of these datasets.

Finding all minimal hitting sets is an NP-complete problem [11]. Since learning a DAG structure from data is also an NP-complete problem [12], the ION algorithm, as given above, requires a superexponential (in $\mathcal{V}$) number of operations and is often computationally intractable even for small sizes of $|\mathcal{V}|$. In practice, however, we can break the minimal hitting set problem into a sequence of smaller subproblems and use a branch and bound approach that is tractable in many cases and still results in an asymptotically correct algorithm. We tested several such strategies. The method which most effectively balanced time and space complexity tradeoffs was to first find all minimal hitting sets which make all possibly active trails of length 2 that correspond to d-separations in some $\mathcal{G}_i \in \mathbf{G}$ not active, then find the structures resulting from making and propagating these changes that are consistent with every $\mathcal{G}_i \in \mathbf{G}$, and iteratively do the same for each of these structures, increasing the length of possibly active trails considered until trails of all sizes are considered.

## 4 Experimental results

We first used synthetic data to evaluate the performance of ION with known ground truth. In the first experiment, we generated 100 random 4-node DAGs using the MCMC algorithm described in [13] with random discrete parameters (conditional probability tables for the factors in the decomposition shown in section 2). For each DAG, we then randomly chose two subsets of size 2 or 3 of the nodes in the DAG such that the union of the subsets included all 4 nodes and at least one overlapping variable between the two subsets was present. We used forward sampling to generate two i.i.d. samples of sizes $N = 50$, $N = 100$, $N = 500$, $N = 1000$ and $N = 2500$ from the DAGs for only the variables in each subset. We used both FCI and GES with latent variable postprocessing to

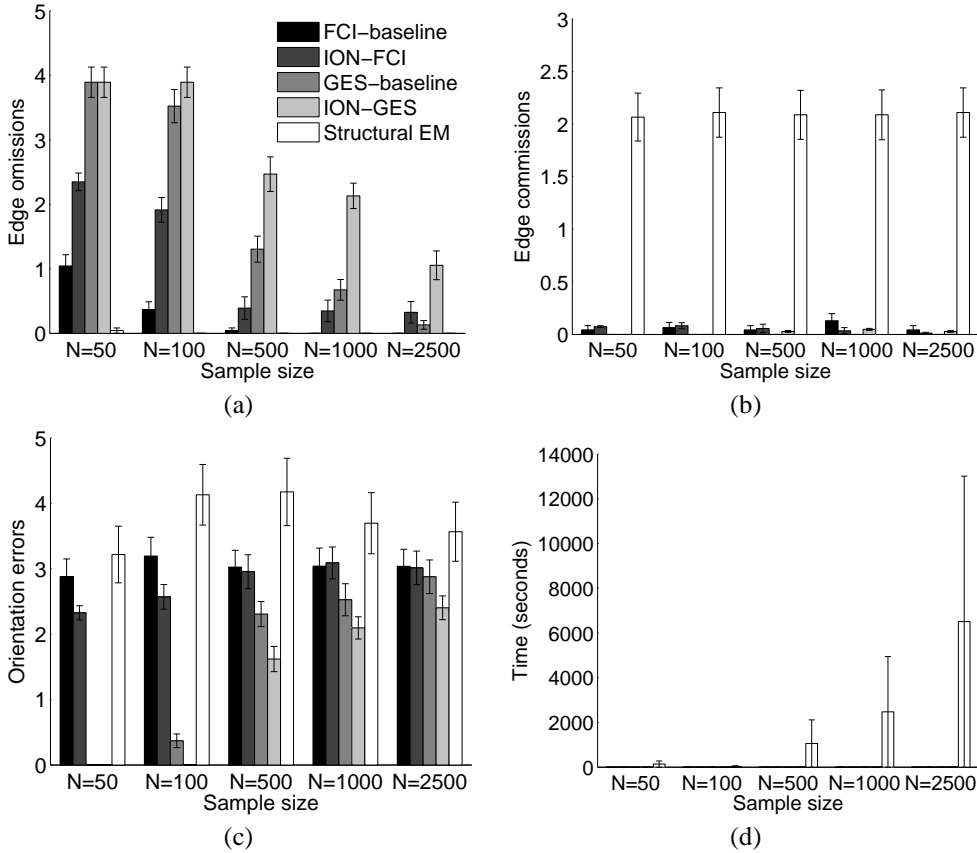

Figure 1: (a) edge omissions, (b) edge commissions, (c) orientation errors, and (d) runtimes

generate PAGs for each of these samples which were input to ION. To evaluate the accuracy of ION, we counted the number of edge omission, edge commision, and orientation errors (▶ instead of −) for each PAG in the ION output set and averaged the results. These results were then averaged across all of the 100 4-node structures. Figure 1 shows the averaged results for these methods along with 3 other methods we included for comparison. *ION-FCI* and *ION-GES* refer the the performance of ION when the input PAGs are obtained using the FCI algorithm and the GES algorithm with latent variable postprocessing, respectively. For *Structural EM*, we took each of the datasets over subsets of the nodes in each DAG and formed a concatenated dataset, as described in section 1, which was input to the Structural EM algorithm.[2] For *FCI-baseline* and *GES-baseline*, we used forward sampling to generate another i.i.d. sample of sizes $N = 50$, $N = 100$, $N = 500$, $N = 1000$ and $N = 2500$ for all of the variables in each DAG and used these datasets as input for the FCI and GES with latent variable postprocessing algorithms, respectively, to obtain a measure for how well these algorithms perform when no data is missing. The average runtimes for each method are also reported in figure 1. Error bars show $95\%$ confidence intervals. We first note the performance of Structural EM. Almost no edge omission errors are made, but more edge commissions errors are made than any of the other methods and the edge commission errors do not decrease as the sample size increases. When we looked at the results, we found that Structural EM always returned either the complete graph or a graph that was almost complete, indicating that Structural EM is not a reliable method for causal discovery in this scenario where there is a highly structured pattern to the missing data. Furthermore, the runtime for Structural EM was considerably higher than any of the other methods. For the larger sample sizes (where more missing values need to be estimated at each iteration), a single run required several hours in some instances. Due to its significant computation time, we

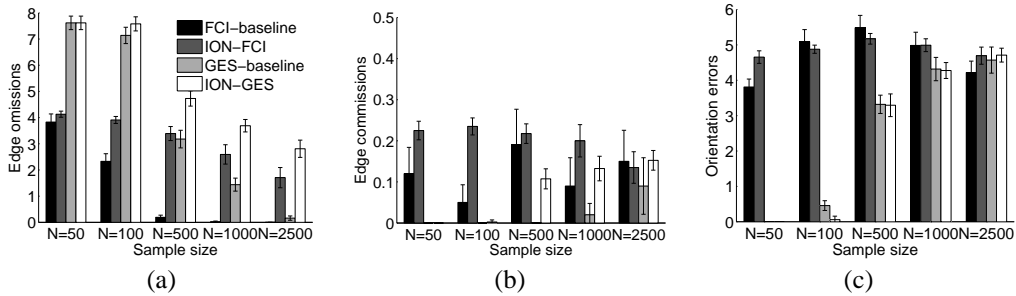

Figure 2: (a) edge omissions, (b) edge comissions, and (c) orientation errors

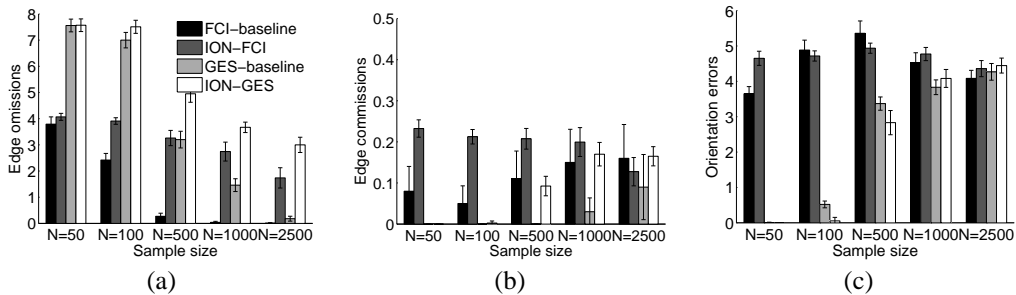

Figure 3: (a) edge omissions, (b) edge comissions, and (c) orientation errors

were unable to use Structural EM with larger DAG structures so it is excluded in the experiments below. The FCI-baseline and GES-baseline methods performed similarly to previous simulations of them. The ION-FCI and ION-GES methods performed similarly to the FCI-baseline and GES-baseline methods but made slightly more errors and showed slower convergence (due to the missing data). Very few edge commission errors were made. Slightly more edge omission errors were made, but these errors decrease as the sample size increases. Some edge orientation errors were made even for the larger sample sizes. This is due to the fact that each of the algorithms returns an equivalence class of DAGs rather than a single DAG. Even if the correct equivalence class is discovered, errors result after comparing the ground truth DAG to every DAG in the equivalence class and averaging. We also note that there are fewer orientation errors for the GES-baseline and ION-GES methods on the two smallest sample sizes than all of the other sample sizes. While this may seem surprising, it is simply a result of the fact that more edge omission errors are made for these cases.

We repeated the above experiment for 3 similar cases where we used 6-node DAG structures rather than 4-node DAG structures: (i) two i.i.d. samples were generated for random subsets of sizes 2-5 with only 1 variable that is not overlapping between the two subsets; (ii) two i.i.d. samples were generated for random subsets of sizes 2-5 with only 2 variables that are not overlapping between the two subsets; (iii) three i.i.d. samples were generated for random subsets of sizes 2-5 with only 1 variable that is not overlapping between any pair of subsets. Figures 2, 3, and 4 show edge omission, edge commission, and orientation errors for each of these cases, respectively. In general, the performance in each case is similar to the performance for the 4-node case.

We also tested the performance of ION-FCI using a real world dataset measuring IQ and various neuroanatomical and other traits [14]. We divided the variables into two subsets with overlapping variables based on domain grounds: (a) variables that might be included in a study on the relationship between neuroanatomical traits and IQ; and (b) variables for a study on the relationship between IQ, sex, and genotype, with brain volume and head circumference included as possible confounders. Figures 5a and 5b show the FCI output PAGs when only the data for each of these subsets of the variables is provided as input, respectively. Figure 5c shows the output PAG of ION-FCI when these two resulting PAGs are used as input. We also ran FCI on the complete dataset to have a comparison. Figure 5d shows this PAG.

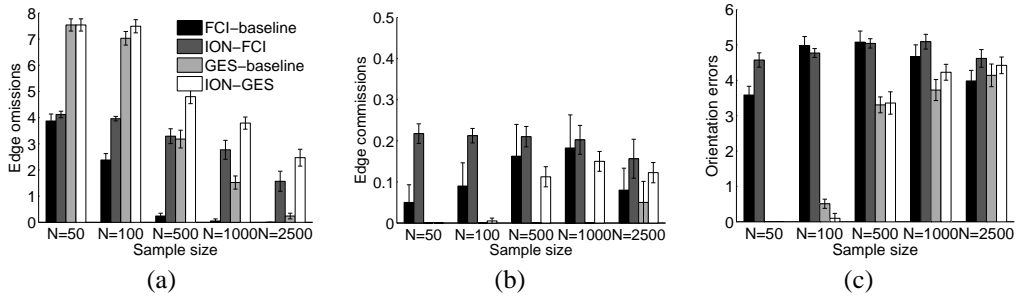

Figure 4: (a) edge omissions, (b) edge comissions, and (c) orientation errors

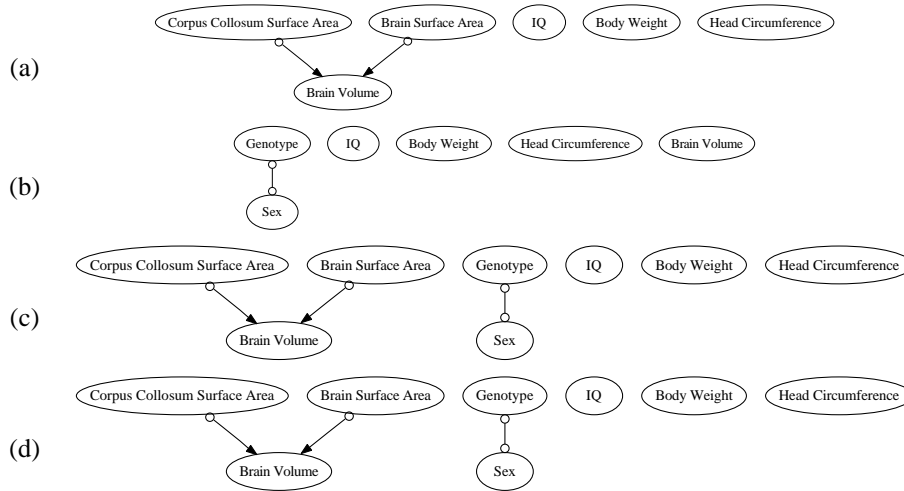

Figure 5: (a) FCI output PAG for variables in subset a, (b) FCI output PAG for variables in subset b, (c) ION output PAG when using the FCI ouput PAGs for variables in subset a and variables in subset b as input, and (d) FCI output PAG for all variables

In this particular case, the output of ION-FCI consists of only a single PAG, which is identical to the result when FCI is given the complete dataset as input. This case shows that in some instances, ION-FCI can recover as much information about the true DAG structure as FCI even when less information can be extracted from the ION-FCI input. We note that the graphical structure of the complete PAG (figures 5c and 5d) is the union of the structures shown in figures 5a and 5b. While visually this may appear to be a trivial example for ION where all of the relevant information can be extracted in the first steps, there is in fact much processing required in later stages in the algorithm to determine the structure around the nonoverlapping variables.

## 5  Conclusions

In practice, researchers are often unable to find or construct a single, complete dataset containing every variable they may be interested in (or doing so is very costly). We thus need some way of integrating information about causal relationships that can be discovered from a collection of datasets with related variables [5]. Standard causal discovery algorithms cannot be used, since they take only a single dataset as input. To address this open problem, we proposed the ION algorithm, an asymptotically correct algorithm for discovering the complete set of causal DAG structures that are consistent with such data.

While the results presented in section 4 indicate that ION is useful in smaller domains when the branch and bound approach described in section 3 is used, a number of issues must be addressed before ION or a simlar algorithm is useful for higher dimensional datasets. Probably the most significant problem is resolving contradictory information among overlapping variables in different

input PAGs, i.e. $X$ is a parent of $Y$ in one PAG and a child of $Y$ in another PAG, resulting from statistical errors or when the input samples are not identically distributed. ION currently ignores such information rather than attempting to resolve it. This increases uncertainty and thus the size of the resulting output set of PAGs. Furthermore, simply ignoring such information does not always avoid conflicts. In some of such cases, ION will not discover any PAGs which entail the correct d-separations and d-connections. Thus, no output PAGs are returned. When performing conditional independence tests or evaluating score functions, statistical errors occur more frequently as the dimensionality of a dataset increases, unless the sample size also increases at an exponential rate (resulting from the so-called *curse of dimensionality*). Thus, until reliable methods for resolving conflicting information from input PAGs are developed, ION and similar algorithms will not in general be useful for higher dimensional datasets. Furthermore, while the branch and bound approach described in section 3 is a significant improvement over other methods we tested for computing minimal hitting sets, its memory requirements are still considerable in some instances. Other algorithmic strategies should be explored in future research.

## Acknowledgements

We thank Joseph Ramsey, Peter Spirtes, and Jiji Zhang for helpful discussions and pointers. We thank Frank Wimberley for implementing the version of Structural EM we used. R.E.T. was supported by the James S. McDonnell Foundation Causal Learning Collaborative Initiative. C.G. was supported by a grant from the James S. McDonnell Foundation.

## Footnotes

[1]We use the standard GES algorithm to learn a DAG structure from the data and then use the FCI rules to check for possible latent common causes.

[2]We ran Structural EM with 5 random restarts and chose the model with the highest BDeu score to avoid converging to local maxima. Random "chains" of nodes were used as the initial models. Structural EM was never stopped before convergence.

## References

[1] P. Spirtes, C. Glymour, and R. Scheines. *Causation, Prediction, and Search*. MIT Press, 2nd edition, 2000.

[2] J. Pearl. *Causality: Models, Reasoning, and Inference*. Cambridge University Press, 2000.

[3] D. M. Chickering. Optimal structure identification with greedy search. *Journal of Machine Learning Research*, 3:507–554, 2002.

[4] D. Danks. Learning the causal structure of overlapping variable sets. In *Discovery Science: Proceedings of the 5th International Conference*, 2002.

[5] D. Danks. Scientific coherence and the fusion of experimental results. *The British Journal for the Philosophy of Science*, 56:791–807, 2005.

[6] S. Rässler. *Statistical Matching*. Springer, 2002.

[7] D. B. Rubin. *Multiple Imputation for Nonresponse in Surveys*. Wiley & Sons, 1987.

[8] N. Friedman. The Bayesian structural EM algorithm. In *Proceedings of the 14th Conference on Uncertainty in Artificial Intelligence*, 1998.

[9] J. Pearl. *Probabilistic Reasoning in Intelligent Systems: Networks of Plausible Inference*. Morgan Kauffmann Publishers, 1988.

[10] J. Zhang. A characterization of markov equivalence classes for causal models with latent variables. In *Proceedings of the 23rd Conference on Uncertainty in Artificial Intelligence*, 2007.

[11] R. Greiner, B. A. Smith, and R. W. Wilkerson. A correction to the algorithm in Reiter's theory of diagnosis. *Artificial Intelligence*, 41:79–88, 1989.

[12] D. M. Chickering. Learning Bayesian networks is NP-complete. In *Proceedings of the 5th International Workshop on Artificial Intelligence and Statistics*, 1995.

[13] G. Melançon, I. Dutour, and M. Bousquet-Mélou. Random generation of dags for graph drawing. Technical Report INS-R0005, Centre for Mathematics and Computer Sciences, Amsterdam, 2000.

[14] M. J. Tramo, W. C. Loftus, R. L Green, T. A. Stukel, J. B. Weaver, and M. S. Gazzaniga. Brain size, head size, and IQ in monozygotic twins. *Neurology*, 50:1246–1252, 1998.

